# Hierarchical Recurrent Neural Networks for Long-Term Dependencies

**Salah El Hihi**
Dept. Informatique et
Recherche Opérationnelle
Université de Montréal
Montreal, Qc H3C-3J7
elhihi@iro.umontreal.ca

**Yoshua Bengio** *
Dept. Informatique et
Recherche Opérationnelle
Université de Montréal
Montreal, Qc H3C-3J7
bengioy@iro.umontreal.ca

## Abstract

We have already shown that extracting long-term dependencies from sequential data is difficult, both for deterministic dynamical systems such as recurrent networks, and probabilistic models such as hidden Markov models (HMMs) or input/output hidden Markov models (IOHMMs). In practice, to avoid this problem, researchers have used domain specific a-priori knowledge to give meaning to the hidden or state variables representing past context. In this paper, we propose to use a more general type of a-priori knowledge, namely that the temporal dependencies are structured hierarchically. This implies that long-term dependencies are represented by variables with a long time scale. This principle is applied to a recurrent network which includes delays and multiple time scales. Experiments confirm the advantages of such structures. A similar approach is proposed for HMMs and IOHMMs.

## 1 Introduction

Learning from examples basically amounts to identifying the relations between random variables of interest. Several learning problems involve *sequential data*, in which the variables are ordered (e.g., time series). Many learning algorithms take advantage of this sequential structure by assuming some kind of homogeneity or continuity of the model over time, e.g., by sharing parameters for different times, as in Time-Delay Neural Networks (TDNNs) (Lang, Waibel and Hinton, 1990), recurrent neural networks (Rumelhart, Hinton and Williams, 1986), or hidden Markov models (Rabiner and Juang, 1986). This general a-priori assumption considerably simplifies the learning problem.

In previous papers (Bengio, Simard and Frasconi, 1994; Bengio and Frasconi, 1995a), we have shown for recurrent networks and Markovian models that, even with this assumption, dependencies that span longer intervals are significantly harder to learn. In all of the systems we have considered for learning from sequential data, some form of representation of context (or state) is required (to summarize all "useful" past information). The "hard learning" problem is to *learn to represent context*, which involves performing the proper

*credit assignment* through time. Indeed, in practice, recurrent networks (e.g., injecting prior knowledge for grammar inference (Giles and Omlin, 1992; Frasconi et al., 1993)) and HMMs (e.g., for speech recognition (Levinson, Rabiner and Sondhi, 1983; Rabiner and Juang, 1986)) work quite well when the representation of context (the meaning of the state variable) is decided a-priori. The hidden variable is not any more completely hidden. Learning becomes much easier. Unfortunately, this requires a very precise knowledge of the appropriate state variables, which is not available in many applications.

We have seen that the successes of TDNNs, recurrent networks and HMMs are based on a general assumption on the sequential nature of the data. In this paper, we propose another, simple, a-priori assumption on the sequences to be analyzed: the temporal dependencies have a hierarchical structure. This implies that dependencies spanning long intervals are "robust" to small local changes in the timing of events, whereas dependencies spanning short intervals are allowed to be more sensitive to the precise timing of events. This yields a multi-resolution representation of state information. This general idea is not new and can be found in various approaches to learning and artificial intelligence. For example, in convolutional neural networks, both for sequential data with TDNNs (Lang, Waibel and Hinton, 1990), and for 2-dimensional data with MLCNNs (LeCun et al., 1989; Bengio, LeCun and Henderson, 1994), the network is organized in layers representing features of increasing temporal or spatial coarseness. Similarly, mostly as a tool for analyzing and preprocessing sequential or spatial data, wavelet transforms (Daubechies, 1990) also represent such information at multiple resolutions. Multi-scale representations have also been proposed to improve reinforcement learning systems (Singh, 1992; Dayan and Hinton, 1993; Sutton, 1995) and path planning systems. However, with these algorithms, one generally assumes that the state of the system is observed, whereas, in this paper we concentrate on the difficulty of learning what the state variable should represent. A related idea using a hierarchical structure was presented in (Schmidhuber, 1992).

On the HMM side, several researchers (Brugnara et al., 1992; Suaudeau, 1994) have attempted to improve HMMs for speech recognition to better model the different types of variables, intrinsically varying at different time scales in speech. In those papers, the focus was on setting an a-priori representation, not on learning how to represent context.

In section 2, we attempt to draw a common conclusion from the analyses performed on recurrent networks and HMMs to learn to represent long-term dependencies. This will justify the proposed approach, presented in section 3. In section 4 a specific hierarchical model is proposed for recurrent networks, using different time scales for different layers of the network. Experiments performed with this model are described in section 4. Finally, we discuss a similar scheme for HMMs and IOHMMs in section 5.

## 2   Too Many Products

In this section, we take another look at the analyses of (Bengio, Simard and Frasconi, 1994) and (Bengio and Frasconi, 1995a), for recurrent networks and HMMs respectively. The objective is to draw a parallel between the problems encountered with the two approaches, in order to guide us towards some form of solution, and justify the proposals made here. First, let us consider the deterministic dynamical systems (Bengio, Simard and Frasconi, 1994) (such as recurrent networks), which map an input sequence $u_1, \ldots, u_T$ to an output sequence $\hat{y}_1, \ldots, \hat{y}_T$. The state or context information is represented at each time $t$ by a variable $x_t$, for example the activities of all the hidden units of a recurrent network:

$$x_t = f(x_{t-1}, u_t) \tag{1}$$

where $u_t$ is the system input at time $t$ and $f$ is a differentiable function (such as $\tanh(Wx_{t-1} + u_t)$). When the sequence of inputs $u_1, u_2, \ldots, u_T$ is given, we can write $x_t = f_t(x_{t-1}) = f_t(f_{t-1}(\ldots f_1(x_0))\ldots)$. A learning criterion $C_t$ yields gradients on outputs, and therefore on the state variables $x_t$. Since parameters are shared across time, learning using a gradient-based algorithm depends on the influence of parameters $W$ on $C_t$ through all time steps before $t$:

$$\frac{\partial C_t}{\partial W} = \sum_{\tau} \frac{\partial C_t}{\partial x_t} \frac{\partial x_t}{\partial x_\tau} \frac{\partial x_\tau}{\partial W} \tag{2}$$

The Jacobian matrix of derivatives $\frac{\partial x_t}{\partial x_\tau}$ can further be factored as follows:

$$\frac{\partial x_t}{\partial x_\tau} = \frac{\partial x_t}{\partial x_{t-1}} \frac{\partial x_{t-1}}{\partial x_{t-2}} \cdots \frac{\partial x_{\tau+1}}{\partial x_\tau} = f_t' f_{t-1}' \cdots f_{\tau+1}' \tag{3}$$

Our earlier analysis (Bengio, Simard and Frasconi, 1994) shows that the difficulty revolves around the matrix product in equation 3. In order to reliably "store" information in the dynamics of the network, the state variable $x_t$ must remain in regions where $|f_t'| < 1$ (i.e., near enough to a stable attractor representing the stored information). However, the above products then rapidly converge to 0 when $t - \tau$ increases. Consequently, the sum in 2 is dominated by terms corresponding to short-term dependencies ($t - \tau$ is small).

Let us now consider the case of Markovian models (including HMMs and IOHMMs (Bengio and Frasconi, 1995b)). These are probabilistic models, either of an "output" sequence $P(y_1 \ldots y_T)$ (HMMs) or of an output sequence given an input sequence $P(y_1 \ldots y_T | u_1 \ldots u_T)$ (IOHMMs). Introducing a discrete state variable $x_t$ and using Markovian assumptions of independence this probability can be factored in terms of transition probabilities $P(x_t|x_{t-1})$ (or $P(x_t|x_{t-1}, u_t)$) and output probabilities $P(y_t|x_t)$ (or $P(y_t|x_t, u_t)$). According to the model, the distribution of the state $x_t$ at time $t$ given the state $x_\tau$ at an earlier time $\tau$ is given by the matrix

$$P(x_t|x_\tau) = P(x_t|x_{t-1})P(x_{t-1}|x_{t-2}) \ldots P(x_{\tau+1}|x_\tau) \tag{4}$$

where each of the factors is a matrix of transition probabilities (conditioned on inputs in the case of IOHMMs). Our earlier analysis (Bengio and Frasconi, 1995a) shows that the difficulty in representing and learning to represent context (i.e., learning what $x_t$ should represent) revolves around equation 4. The matrices in the above equations have one eigenvalue equal to 1 (because of the normalization constraint) and the others $\leq 1$. In the case in which all eigenvalues are 1 the matrices have only 1's and 0's, i.e, we obtain deterministic dynamics for IOHMMs or pure cycles for HMMs (which cannot be used to model most interesting sequences). Otherwise the above product converges to a lower rank matrix (some or most of the eigenvalues converge toward 0). Consequently, $P(x_t|x_\tau)$ becomes more and more independent of $x_\tau$ as $t - \tau$ increases. Therefore, **both** representing **and** learning context becomes more difficult as the span of dependencies increases or when the Markov model is more non-deterministic (transition probabilities not close to 0 or 1).

Clearly, a common trait of both analyses lies in taking *too many products, too many time steps, or too many transformations* to relate the state variable at time $\tau$ with the state variable at time $t > \tau$, as in equations 3 and 4. Therefore the idea presented in the next section is centered on allowing *several paths* between $x_\tau$ and $x_t$, some with few "transformations" and some with many transformations. At least through those with few transformations, we expect context information (forward), and credit assignment (backward) to propagate more easily over longer time spans than through "paths" involving many transformations.

## 3 Hierarchical Sequential Models

Inspired by the above analysis we introduce an assumption about the sequential data to be modeled, although it will be a very simple and general a-priori on the structure of the data. Basically, we will assume that the sequential structure of data can be described *hierarchically*: long-term dependencies (e.g., between two events remote from each other in time) do not depend on a precise time scale (i.e., on the precise timing of these events). Consequently, in order to represent a context variable taking these long-term dependencies into account, we will be able to use a coarse time scale (or a slowly changing state variable).

Therefore, instead of a single homogeneous state variable, we will introduce several levels of state variables, each "working" at a different time scale. To implement in a discrete-time system such a multi-resolution representation of context, two basic approaches can be considered. Either the higher level state variables change value less often or they are constrained to change more slowly at each time step. In our experiments, we have considered input and output variables both at the shortest time scale (highest frequency), but one of the potential advantages of the approach presented here is that it becomes very

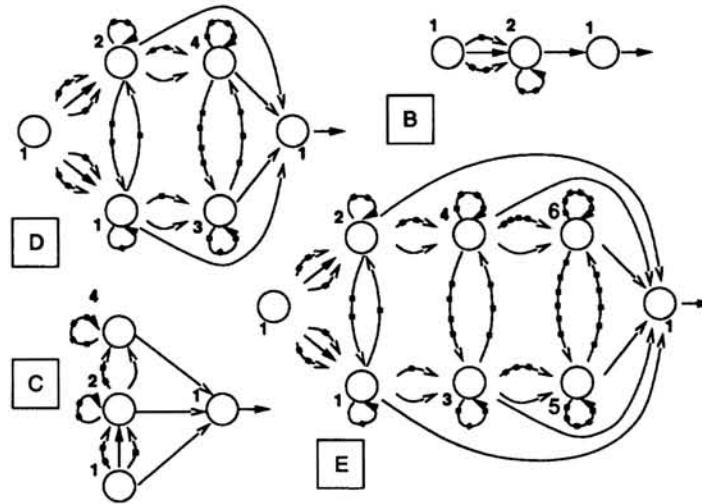

Figure 1: Four multi-resolution recurrent architectures used in the experiments. Small squares represent a discrete delay, and numbers near each neuron represent its time scale. The architectures **B** to **E** have respectively 2, 3, 4, and 6 time scales.

simple to incorporate input and output variables that operate at different time scales. For example, in speech recognition and synthesis, the variable of interest is not only the speech signal itself (fast) but also slower-varying variables such as prosodic (average energy, pitch, etc...) and phonemic (place of articulation, phoneme duration) variables. Another example is in the application of learning algorithms to financial and economic forecasting and decision taking. Some of the variables of interest are given daily, others weekly, monthly, etc...

## 4   Hierarchical Recurrent Neural Network: Experiments

As in TDNNs (Lang, Waibel and Hinton, 1990) and reverse-TDNNs (Simard and LeCun, 1992), we will use discrete time delays and subsampling (or oversampling) in order to implement the multiple time scales. In the time-unfolded network, paths going through the recurrences in the slow varying units (long time scale) will carry context farther, while paths going through faster varying units (short time scale) will respond faster to changes in input or desired changes in output. Examples of such multi-resolution recurrent neural networks are shown in Figure 1. Two sets of simple experiments were performed to validate some of the ideas presented in this paper. In both cases, we compare a hierarchical recurrent network with a single-scale fully-connected recurrent network.

In the first set of experiments, we want to evaluate the performance of a hierarchical recurrent network on a problem already used for studying the difficulty in learning long-term dependencies (Bengio, Simard and Frasconi, 1994; Bengio and Frasconi, 1994). In this 2-class problem, the network has to detect a pattern at the beginning of the sequence, keeping a bit of information in "memory" (while the inputs are noisy) until the end of the sequence (supervision is only a the end of the sequence). As in (Bengio, Simard and Frasconi, 1994; Bengio and Frasconi, 1994) only the first 3 time steps contain information about the class (a 3-number pattern was randomly chosen for each class within $[-1, 1]^3$). The length of the sequences is varied to evaluate the effect of the span of input/output dependencies. Uniformly distributed noisy inputs between -.1 and .1 are added to the initial patterns as well as to the remainder of the sequence. For each sequence length, 10 trials were run with different initial weights and noise patterns, with 30 training sequences. Experiments were performed with sequence of lengths 10, 20, 40 and 100.

Several recurrent network architectures were compared. All were trained with the same algorithm (back-propagation through time) to minimize the sum of squared differences between the final output and a desired value. The simplest architecture (**A**) is similar to architecture **B** in Figure 1 but it is not hierarchical: it has a single time scale. Like the

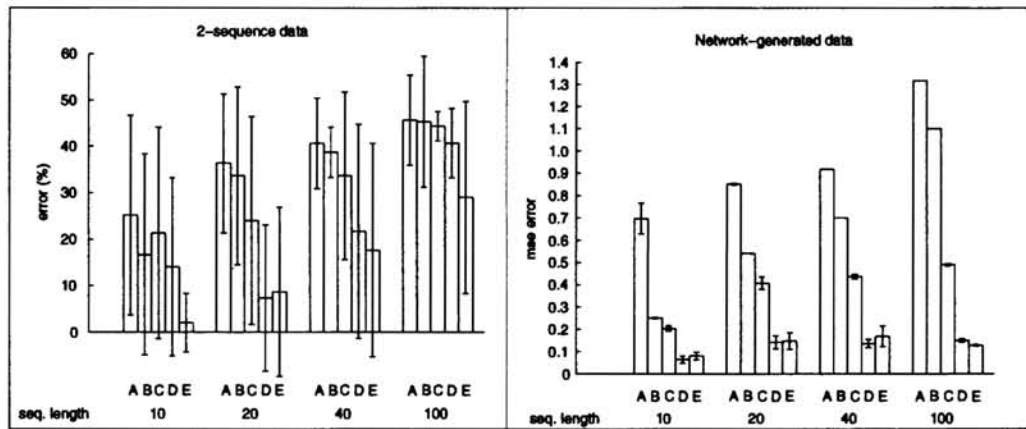

Figure 2: Average classification error after training for 2-sequence problem (left, classification error) and network-generated data (right, mean squared error), for varying sequence lengths and architectures. Each set of 5 consecutive bars represents the performance of 5 architectures **A** to **E**, with respectively 1, 2, 3, 4 and 6 time scales (the architectures **B** to **E** are shown in Figure 1). Error bars show the standard deviation over 10 trials.

other networks, it has however a theoretically "sufficient" architecture, i.e., there exists a set of weights for which it classifies perfectly the training sequences. Four of the five architectures that we compared are shown in Figure 1, with an increasing number of levels in the hierarchy. The performance of these four architectures (**B** to **E**) as well as the architecture with a single time-scale (**A**) are compared in Figure 2 (left, for the 2-sequence problem). Clearly, adding more levels to the hierarchy has significantly helped to reduce the difficulty in learning long-term dependencies.

In a second set of experiments, a hierarchical recurrent network with 4 time scales was initialized with random (but large) weights and used to generate a data set. To generate the inputs as well as the outputs, the network has feedback links from hidden to input units. At the initial time step as well as at 5% of the time steps (chosen randomly), the input was clamped with random values to introduce some further variability. It is a regression task, and the mean squared error is shown on Figure 2. Because of the network structure, we expect the data to contain long-term dependencies that can be modeled with a hierarchical structure. 100 training sequences of length 10, 20, 40 and 100 were generated by this network. The same 5 network architectures as in the previous experiments were compared (see Figure 1 for architectures **B** to **E**), with 10 training trials per network and per sequence length. The results are summarized in Figure 2 (right). More high-level hierarchical structure appears to have improved performance for long-term dependencies. The fact that the simpler 1-level network does not achieve a good performance suggests that there were some difficult long-term dependencies in the the artificially generated data set. It is interesting to compare those results with those reported in (Lin et al., 1995) which show that using longer delays in certain recurrent connections helps learning longer-term dependencies. In both cases we find that introducing longer time scales allows to learn dependencies whose span is proportionally longer.

## 5    Hierarchical HMMs

How do we represent multiple time scales with a HMM? Some solutions have already been proposed in the speech recognition literature, motivated by the obvious presence of different time scales in the speech phenomena. In (Brugnara et al., 1992) two Markov chains are coupled in a "master/slave" configuration. For the "master" HMM, the observations are slowly varying features (such as the signal energy), whereas for the "slave" HMM the observations are the speech spectra themselves. The two chains are synchronous and operate at the same time scale, therefore the problem of diffusion of credit in HMMs would probably also make difficult the learning of long-term dependencies. Note on the other

hand that in most applications of HMMs to speech recognition the meaning of states is fixed a-priori rather than learned from the data (see (Bengio and Frasconi, 1995a) for a discussion). In a more recent contribution, Nelly Suaudeau (Suaudeau, 1994) proposes a "two-level HMM" in which the higher level HMM represents "segmental" variables (such as phoneme duration). The two levels operate at different scales: the higher level state variable represents the phonetic identity and models the distributions of the average energy and the duration within each phoneme. Again, this work is not geared towards learning a representation of context, but rather, given the traditional (phoneme-based) representation of context in speech recognition, towards building a better model of the distribution of "slow" segmental variables such as phoneme duration and energy. Another promising approach was recently proposed in (Saul and Jordan, 1995). Using decimation techniques from statistical mechanics, a polynomial-time algorithm is derived for parallel Boltzmann chains (which are similar to parallel HMMs), which can operate at different time scales.

The ideas presented here point toward a HMM or IOHMM in which the (hidden) state variable $x_t$ is represented by the Cartesian product of several state variables $x_t^s$, each "working" at a different time scale: $x_t = (x_t^1, x_t^2, \ldots, x_t^L)$. To take advantage of the decomposition, we propose to consider that the state distributions at the different levels are conditionally independent (given the state at the previous time step and at the current and previous levels). Transition probabilities are therefore factored as followed:

$$P(x_t|x_{t-1}) = \prod_s P(x_t^s|x_{t-1}^s, x_{t-1}^{s-1}) \tag{5}$$

To force the state variable at a each level to effectively work at a given time scale, self-transition probabilities are constrained as follows (using above independence assumptions):

$$P(x_t^s{=}j_s|x_{t-1}^1{=}j_1, \ldots, x_{t-1}^s{=}j_s, \ldots, x_{t-1}^L{=}j_S) = P(x_t^s{=}j_s|x_{t-1}^s{=}j_s, x_{t-1}^{s-1}{=}j_{s-1}) = w_s$$

## 6  Conclusion

Motivated by the analysis of the problem of learning long-term dependencies in sequential data, i.e., of learning to represent context, we have proposed to use a very general assumption on the structure of sequential data to reduce the difficulty of these learning tasks. Following numerous previous work in artificial intelligence we are assuming that context can be represented with a hierarchical structure. More precisely, here, it means that long-term dependencies are insensitive to small timing variations, i.e., they can be represented with a coarse temporal scale. This scheme allows context information and credit information to be respectively propagated forward and backward more easily.

Following this intuitive idea, we have proposed to use *hierarchical recurrent networks* for sequence processing. These networks use multiple-time scales to achieve a multi-resolution representation of context. Series of experiments on artificial data have confirmed the advantages of imposing such structures on the network architecture. Finally we have proposed a similar application of this concept to hidden Markov models (for density estimation) and input/output hidden Markov models (for classification and regression).

## Footnotes

*also, AT&T Bell Labs, Holmdel, NJ 07733

## References

Bengio, Y. and Frasconi, P. (1994). Credit assignment through time: Alternatives to backpropagation. In Cowan, J., Tesauro, G., and Alspector, J., editors, *Advances in Neural Information Processing Systems 6*. Morgan Kaufmann.

Bengio, Y. and Frasconi, P. (1995a). Diffusion of context and credit information in markovian models. *Journal of Artificial Intelligence Research*, 3:223–244.

Bengio, Y. and Frasconi, P. (1995b). An input/output HMM architecture. In Tesauro, G., Touretzky, D., and Leen, T., editors, *Advances in Neural Information Processing Systems 7*, pages 427–434. MIT Press, Cambridge, MA.

Bengio, Y., LeCun, Y., and Henderson, D. (1994). Globally trained handwritten word recognizer using spatial representation, space displacement neural networks and hidden Markov models. In Cowan, J., Tesauro, G., and Alspector, J., editors, *Advances in Neural Information Processing Systems 6*, pages 937–944.

Bengio, Y., Simard, P., and Frasconi, P. (1994). Learning long-term dependencies with gradient descent is difficult. *IEEE Transactions on Neural Networks*, 5(2):157–166.

Brugnara, F., DeMori, R., Giuliani, D., and Omologo, M. (1992). A family of parallel hidden markov models. In *International Conference on Acoustics, Speech and Signal Processing*, pages 377–370, New York, NY, USA. IEEE.

Daubechies, I. (1990). The wavelet transform, time-frequency localization and signal analysis. *IEEE Transaction on Information Theory*, 36(5):961–1005.

Dayan, P. and Hinton, G. (1993). Feudal reinforcement learning. In Hanson, S. J., Cowan, J. D., and Giles, C. L., editors, *Advances in Neural Information Processing Systems 5*, San Mateo, CA. Morgan Kaufmann.

Frasconi, P., Gori, M., Maggini, M., and Soda, G. (1993). Unified integration of explicit rules and learning by example in recurrent networks. *IEEE Transactions on Knowledge and Data Engineering*. (in press).

Giles, C. L. and Omlin, C. W. (1992). Inserting rules into recurrent neural networks. In Kung, Fallside, Sorenson, and Kamm, editors, *Neural Networks for Signal Processing II, Proceedings of the 1992 IEEE workshop*, pages 13–22. IEEE Press.

Lang, K. J., Waibel, A. H., and Hinton, G. E. (1990). A time-delay neural network architecture for isolated word recognition. *Neural Networks*, 3:23–43.

LeCun, Y., Boser, B., Denker, J., Henderson, D., Howard, R., Hubbard, W., and Jackel, L. (1989). Backpropagation applied to handwritten zip code recognition. *Neural Computation*, 1:541–551.

Levinson, S., Rabiner, L., and Sondhi, M. (1983). An introduction to the application of the theory of probabilistic functions of a Markov process to automatic speech recognition. *Bell System Technical Journal*, 64(4):1035–1074.

Lin, T., Horne, B., Tino, P., and Giles, C. (1995). Learning long-term dependencies is not as difficult with NARX recurrent neural networks. Technical Report UMICAS-TR-95-78, Institute for Advanced Computer Studies, University of Mariland.

Rabiner, L. and Juang, B. (1986). An introduction to hidden Markov models. *IEEE ASSP Magazine*, pages 257–285.

Rumelhart, D., Hinton, G., and Williams, R. (1986). Learning internal representations by error propagation. In Rumelhart, D. and McClelland, J., editors, *Parallel Distributed Processing*, volume 1, chapter 8, pages 318–362. MIT Press, Cambridge.

Saul, L. and Jordan, M. (1995). Boltzmann chains and hidden markov models. In Tesauro, G., Touretzky, D., and Leen, T., editors, *Advances in Neural Information Processing Systems 7*, pages 435–442. MIT Press, Cambridge, MA.

Schmidhuber, J. (1992). Learning complex, extended sequences using the principle of history compression. *Neural Computation*, 4(2):234–242.

Simard, P. and LeCun, Y. (1992). Reverse TDNN: An architecture for trajectory generation. In Moody, J., Hanson, S., and Lipmann, R., editors, *Advances in Neural Information Processing Systems 4*, pages 579–588, Denver, CO. Morgan Kaufmann, San Mateo.

Singh, S. (1992). Reinforcement learning with a hierarchy of abstract models. In *Proceedings of the 10th National Conference on Artificial Intelligence*, pages 202–207. MIT/AAAI Press.

Suaudeau, N. (1994). *Un modèle probabiliste pour intégrer la dimension temporelle dans un système de reconnaissance automatique de la parole*. PhD thesis, Université de Rennes I, France.

Sutton, R. (1995). TD models: modeling the world at a mixture of time scales. In *Proceedings of the 12th International Conference on Machine Learning*. Morgan Kaufmann.
